# Fixing two weaknesses of the Spectral Method

**Kevin J. Lang**
Yahoo Research
3333 Empire Ave, Burbank, CA 91504
langk@yahoo-inc.com

## Abstract

We discuss two intrinsic weaknesses of the spectral graph partitioning method, both of which have practical consequences. The first is that spectral embeddings tend to hide the best cuts from the commonly used hyperplane rounding method. Rather than cleaning up the resulting sub-optimal cuts with local search, we recommend the adoption of flow-based rounding. The second weakness is that for many "power law" graphs, the spectral method produces cuts that are highly unbalanced, thus decreasing the usefulness of the method for visualization (see figure 4(b)) or as a basis for divide-and-conquer algorithms. These balance problems, which occur even though the spectral method's quotient-style objective function does encourage balance, can be fixed with a stricter balance constraint that turns the spectral mathematical program into an SDP that can be solved for million-node graphs by a method of Burer and Monteiro.

## 1 Background

Graph partitioning is the NP-hard problem of finding a small graph cut subject to the constraint that neither side of the resulting partitioning of the nodes is "too small". We will be dealing with several versions: the graph bisection problem, which requires perfect $\frac{1}{2} : \frac{1}{2}$ balance; the $\beta$-balanced cut problem (with $\beta$ a fraction such as $\frac{1}{3}$), which requires at least $\beta : (1 - \beta)$ balance; and the quotient cut problem, which requires the small side to be large enough to "pay for" the edges in the cut. The quotient cut metric is $c/\min(a, b)$, where $c$ is the cutsize and $a$ and $b$ are the sizes of the two sides of the cut. All of the well-known variants of the quotient cut metric (e.g. normalized cut [15]) have similar behavior with respect to the issues discussed in this paper.

The spectral method for graph partitioning was introduced in 1973 by Fiedler and Donath & Hoffman [6]. In the mid-1980's Alon & Milman [1] proved that spectral cuts can be at worst quadratically bad; in the mid 1990's Guattery & Miller [10] proved that this analysis is tight by exhibiting a family of $n$-node graphs whose spectral bisections cut $O(n^{2/3})$ edges versus the optimal $O(n^{1/3})$ edges. On the other hand, Spielman & Teng [16] have proved stronger performance guarantees for the special case of spacelike graphs.

The spectral method can be derived by relaxing a quadratic integer program which encodes the graph bisection problem (see section 3.1). The solution to this relaxation is the "Fiedler vector", or second smallest eigenvector of the graph's discrete Laplacian matrix, whose elements $x_i$ can be interpreted as an embedding of the graph on the line. To obtain a

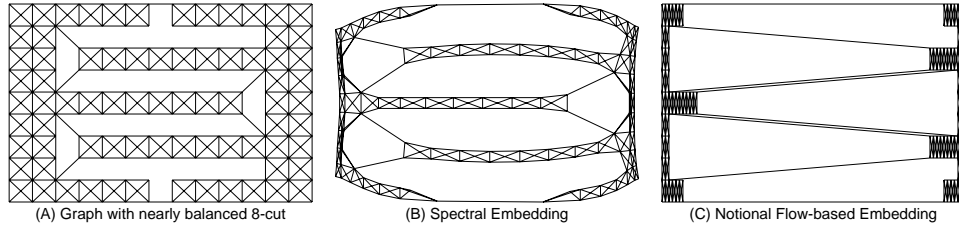

(A) Graph with nearly balanced 8-cut      (B) Spectral Embedding      (C) Notional Flow-based Embedding

Figure 1: The spectral embedding hides the best solution from hyperplane rounding.

specific cut, one must apply a "rounding method" to this embedding. The hyperplane rounding method chooses one of the $n-1$ cuts which separate the nodes whose $x_i$ values lie above and below some split value $\hat{x}$.

## 2   Using flow to find cuts that are hidden from hyperplane rounding

Theorists have long known that the spectral method cannot distinguish between deep cuts and long paths, and that this confusion can cause it to cut a graph in the wrong direction thereby producing the spectral method's worst-case behavior [10]. In this section we will show by example that even when the spectral method is not fooled into cutting in the wrong direction, the resulting embedding can hide the best cuts from the hyperplane rounding method. This is a possible explanation for the frequently made empirical observation (see e.g. [12]) that hyperplane roundings of spectral embeddings are noisy and therefore benefit from cleanup with a local search method such as Fiduccia-Matheyses [8].

Consider the graph in figure 1(a), which has a near-bisection cutting 8 edges. For this graph the spectral method produces the embedding shown in figure 1(b), and recommends that we make a vertical cut (across the horizontal dimension which is based on the Fiedler vector). This is correct in a generalized sense, but it is obvious that no hyperplane (or vertical line in this picture) can possibly extract the optimal 8-edge cut.

Some insight into why spectral embeddings tend to have this problem can be obtained from the spectral method's electrical interpretation. In this view the graph is represented by a resistor network [7]. Current flowing in this network causes voltage drops across the resistors, thus determining the nodes' voltages and hence their positions. When current flows through a long series of resistors, it induces a progressive voltage drop. This is what causes the excessive length of the embeddings of the horizontal girder-like structures which are blocking all vertical hyperplane cuts in figure 1(b).

If the embedding method were somehow not based on current, but rather on flow, which does not distinguish between a pipe and a series of pipes, then the long girders could retract into the two sides of the embedding, as suggested by figure 1(c), and the best cut would be revealed. Because theoretical flow-like embedding methods such as [14] are currently not practical, we point out that in cases like figure 1(b), where the spectral method has not chosen an incorrect direction for the cut, one can use an S-T max flow problem with the flow running in the recommended direction (horizontally for this embedding) to extract the good cut even though it is hidden from all hyperplanes.

We currently use two different flow-based rounding methods. A method called MQI looks for quotient cuts, and is already described in [13]. Another method, that we shall call Midflow, looks for $\beta$-balanced cuts. The input to Midflow is a graph and an ordering of its nodes (obtained e.g. from a spectral embedding or from the projection of any embedding onto a line). We divide the graph's nodes into 3 sets F, L, and U. The sets F and L respectively contain the first $\beta n$ and last $\beta n$ nodes in the ordering, and U contains the remaining

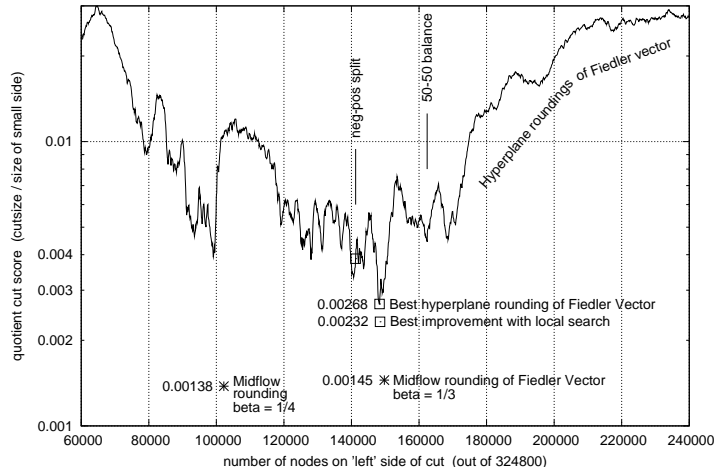

Figure 2: A typical example (see section 2.1) where flow-based rounding beats hyperplane rounding, even when the hyperplane cuts are improved with Fiduccia-Matheyses search. Note that for this spacelike graph, the best quotient cuts have reasonably good balance.

$U = n - 2\beta n$ nodes, which are "up for grabs". We set up an S-T max flow problem with one node for every graph node plus 2 new nodes for the source and sink. For each graph edge there are two arcs, one in each direction, with unit capacity. Finally, the nodes in F are pinned to the source and the nodes in L are pinned to sink by infinite capacity arcs. This max-flow problem can be solved by a good implementation of the push-relabel algorithm (such as Goldberg and Cherkassky's `hi_pr` [4]) in time that empirically is nearly linear with a very good constant factor. Figure 6 shows that solving a MidFlow problem with `hi_pr` can be 1000 times cheaper than finding a spectral embedding with `ARPACK`.

When the goal is finding good $\beta$-balanced cuts, MidFlow rounding is strictly more powerful than hyperplane rounding; from a given node ordering hyperplane rounding chooses the best of $U + 1$ candidate cuts, while MidFlow rounding chooses the best of $2^U$ candidates, including all of those considered by hyperplane rounding. [Similarly, MQI rounding is strictly more powerful than hyperplane rounding for the task of finding good quotient cuts.]

## 2.1   A concrete example

The plot in figure 2 shows a number of cuts in a 324,800 node nearly planar graph derived from a 700x464 pixel downward-looking view of some clouds over some mountains.[1] The y-axis of the plot is quotient cut score; smaller values are better. We note in passing that the commonly used split point $\hat{x} = 0$ does *not* yield the best hyperplane cut. Our main point is that the two cuts generated by MidFlow rounding of the Fiedler vector (with $\beta = \frac{1}{3}$ and $\beta = \frac{1}{4}$) are nearly twice as good as the best hyperplane cut. Even after the best hyperplane cut has been improved by taking the best result of 100 runs of a version of Fiduccia-Matheyses local search, it is still much worse than the cuts obtained by flow-based rounding.

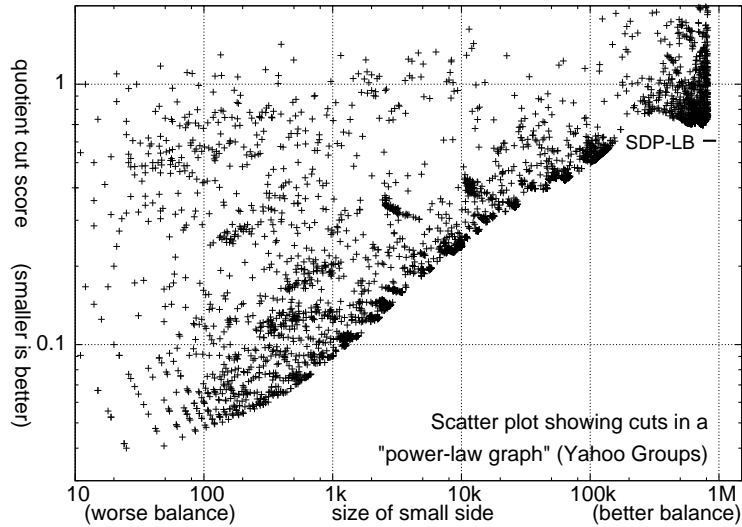

Figure 3: This scatter plot of cuts in a 1.6 million node collaborative filtering graph shows a surprising relationship between cut quality and balance (see section 3). The SDP lower bound proves that *all* balanced cuts are worse than the unbalanced cuts seen on the left.

## 2.2 Effectiveness on real graphs and benchmarks

We have found the flow-based Midflow and MQI rounding methods to be highly effective in practice on diverse classes of graphs including space-like graphs and power law graphs. Results for real-world power law graphs are shown in figure 5. Results for a number of FE meshes can be found on the Graph Partitioning Archive website `http://staffweb.cms.gre.ac.uk/~c.walshaw/partition`, which keeps track of the best nearly balanced cuts ever found for a number of classic benchmarks. Using flow-based rounding to extract cuts from spectral-type embeddings, we have found new record cuts for the majority of the largest graphs on the site, including `fe_body`, `t60k`, `wing`, `brack2`, `fe_tooth`, `fe_rotor`, `598a`, `144`, `wave`, `m14b`, and `auto`. It is interesting to note that the spectral method previously did not own any of the records for these classic benchmarks, although it could have if flow-based rounding had been used instead of hyperplane rounding.

## 3   Finding balanced cuts in "power law" graphs

The spectral method does not require cuts to have perfect balance, but the denominator in its quotient-style objective function does reward balance and punish imbalance. Thus one might expect the spectral method to produce cuts with fairly good balance, and this is what does happen for the class of spacelike graphs that inform much of our intuition.

However, there are now many economically important "power law" [5] graphs whose best quotient cuts have extremely bad balance. Examples at Yahoo include the web graph, social graphs based on DLBP co-authorship and Yahoo IM buddy lists, a music similarity graph, and bipartite collaborative filtering graphs relating Yahoo Groups with users, and advertisers with search phrases. To save space we show one scatter plot (figure 3) of quotient cut scores versus balance that is typical for graphs from this class. We see that apparently there is a tradeoff between these two quantities, and in fact the quotient cut score gets better as

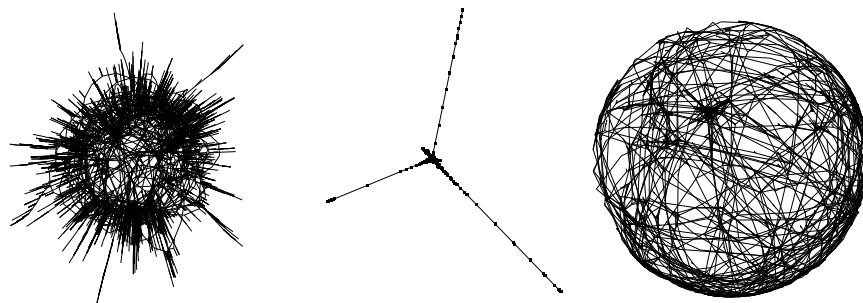

Figure 4: Left: a social graph with octopus structure as predicted by Chung and Lu [5]. Center: a "normalized cut" Spectral embedding chops off one tentacle per dimension. Right: an SDP embedding looks better and is more useful for finding balanced cuts.

balance gets worse, which is exactly the opposite of what one would expect.

When run on graphs of this type, the spectral method (and other quotient cut methods such as Metis+MQI [13]) wants to chop off tiny pieces. This has at least two bad practical effects. First, cutting off a tiny piece after paying for a computation on the whole graph kills the scalability of divide and conquer algorithms by causing their overall run time to increase e.g. from $n \log n$ to $n^2$. Second, low-dimensional spectral embeddings of these graphs (see e.g. figure 4(b) are nearly useless for visualization, and are also very poor inputs for clustering schemes that use a small number of eigenvectors.

These problems can be avoided by solving a semidefinite relaxation of graph bisection that has a much stronger balance constraint. This SDP (explained in the next section) has a long history, with connections to papers going all the way back to Donath and Hoffman [6] (via the concept of "eigenvalue optimization"). In 2004, Arora, Rao, and Vazirani [14] proved the best-ever approximation guarantee for graph partitioning by analysing a version of this SDP which was augmented with certain triangle inequalities that serve much the same purpose as flow (but which are too expensive to solve for large graphs).

### 3.1 A semidefinite program which strengthens the balance requirement

The graph bisection problem can be expressed as a Quadratic Integer Program as follows. There is an $n$-element column vector $x$ of indicator variables $x_i$, each of which assigns one node to a particular side of the cut by assuming a value from the set $\{-1, 1\}$. With these indicator values, the objective function $\frac{1}{4}x^T L x$ (where $L$ is the graph's discrete Laplacian matrix) works out to be equal to the number of edges crossing the cut. Finally, the requirement of perfect balance is expressed by the constraint $x^T e = 0$, where $e$ is a vector of all ones. Since this QIP exactly encodes the graph bisection problem, solving it is NP-hard.

The spectral relaxation of this QIP attains solvability by allowing the indicator variables to assume arbitrary real values, provided that their average squared magnitude is 1.0. After this change, the objective function $\frac{1}{4}x^T L x$ is now just a lower bound on the cutsize. More interestingly for the present discussion, the balance contraint $x^T e = 0$ now permits a qualitatively different kind of balance where a tiny group of nodes moves a long way out from the origin where the nodes acquire enough leverage to counterbalance everyone else. For graphs where the best quotient cut has good balance (e.g. meshes) this does not actually happen, but for graphs whose best quotient cut has bad balance, it does happen, as can be seen in figure 4(b).

These undesired solutions could be ruled out by requiring the squared magnitudes of the indicator values to be 1.0 individually instead of on average. However, in one dimension that would require picking values from the set $\{-1, 1\}$, which would once again cause the problem to be NP-hard. Fortunately, there is a way to escape from this dilemma which was brought to the attention of the CS community by the Max Cut algorithm of Goemans and Williamson [9]: if we allow the indicator variables to assume values that are $r$-dimensional unit vectors for some sufficiently large $r$,[2] then the program is solvable even with the strict requirement that every vector has squared length 1.0. After a small change of notation to reflect the fact that the collected indicator variables now form an $n$ by $r$ matrix $X$ rather than a vector, this idea results in the nonlinear program

$$ min \left\{ \frac{1}{4} L \bullet (XX^T) : diag(XX^T) = e, e^T(XX^T)e = 0 \right\} \qquad (1) $$

which becomes an SDP by a change of variables from $XX^T$ to the "Gram matrix" $G$:

$$ min \left\{ \frac{1}{4} L \bullet G : diag(G) = e, e^T Ge = 0, G \succeq 0 \right\} \qquad (2) $$

The added constraint $G \succeq 0$ requires $G$ to be positive semidefinite, so that it can be factored to get back to the desired matrix of indicator vectors $X$.

### 3.2 Methods for solving the SDP for large graphs

Interior point methods cannot solve (2) for graphs with more than a few thousand nodes, but newer methods achieve better scaling by ensuring that all dense $n$ by $n$ matrices have only an implicit (and approximate) existence. A good example is Helmberg and Rendl's program SBmethod [11], which can solve the dual of (2) for graphs with about 50,000 nodes by converting it to an equivalent "eigenvalue optimization" problem. The output of SBmethod is a low-rank approximate spectral factorization of the Gram matrix, consisting of an estimated rank $r$, plus an $n$ by $r$ matrix $X$ whose rows are the nodes' indicator vectors. SBmethod typically produces $r$-values that are much smaller than $n$ or even $\sqrt{2n}$. Moreover they seem to match the true dimensionality of simple spacelike graphs. For example, for a 3-d mesh we get $r = 4$, which is 3 dimensions for the manifold plus one more dimension for the hypersphere that it is wrapped around.

Burer and Monteiro's direct low-rank solver SDP-LR scales even better [2]. Surprisingly, their approach is to essentially forget about the SDP (2) and instead use non-linear programming techniques to solve (1). Specifically, they use an augmented Lagrangian approach to move the constraints into the objective function, which they then minimize using limited memory BFGS. A follow-up paper [3] provides a theoretical explanation of why the method does not fall into bad local minima despite the apparent non-convexity of (1). We have successfully run Burer and Monteiro's code on large graphs containing more than a million nodes. We typically run it several times with different small fixed values of $r$, and then choose the smallest $r$ which allows the objective function to reach its best known value. On medium-size graphs this produces estimates for $r$ which are in rough agreement with those produced by SBmethod. The run time scaling of SDP-LR is compared with that of ARPACK and hi_pr in figure 6.

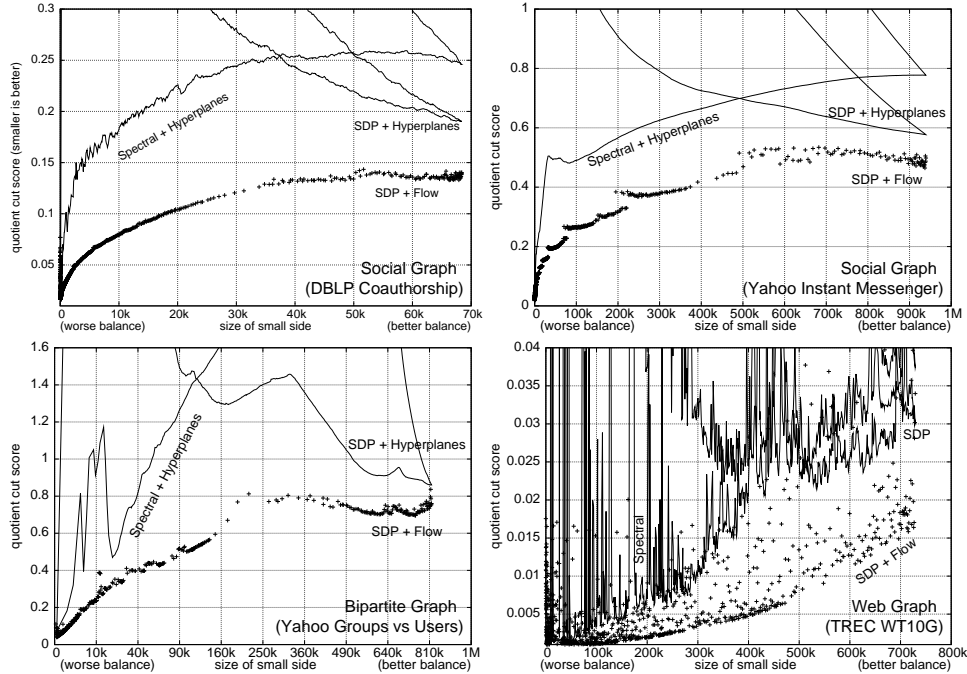

Figure 5: Each of these four plots contains two lines showing the results of sweeping a hyperplane through a spectral embedding and through one dimension of an SDP embedding. In all four cases, the spectral line is lower on the left, and the SDP line is lower on the right, which means that Spectral produces better unbalanced cuts and the SDP produces better balanced cuts. Cuts obtained by rounding random 1-d projections of the SDP embedding using Midflow (to produce $\beta$-balanced cuts) followed by MQI (to improve the quotient cut score) are also shown; these flow-based cuts are consistently better than hyperplane cuts.

## 3.3 Results

We have used the `minbis` program from Burer and Monteiro's `SDP-LR_v0.130301` package (with $r < 10$) to approximately solve (1) for several large graphs including: a 130,000 node social graph representing co-authorship in DBLP; a 1.9 million node social graph built from the buddy lists of a subset of the users of Yahoo Instant Messenger; a 1.6 million node bipartite graph relating Yahoo Groups and users; and a 1.5 million node graph made by symmetrizing the TREC WT10G web graph. It is clear from figure 5 that in all four cases the SDP embedding leads to better balanced cuts, and that flow-based rounding works better hyperplane rounding. Also, figures 4(b) and 4(c) show 3-d Spectral and SDP embeddings of a small subset of the Yahoo IM social graph; the SDP embedding is qualitatively different and arguably better for visualization purposes.

### Acknowledgments

We thank Satish Rao for many useful discussions.

## Footnotes

[1]The graph's edges are unweighted but are chosen by a randomized rule which is more likely to include an edge between two neighboring pixels if they have a similar grey value. Good cuts in the graph tend to run along discontinuities in the image, as one would expect.

[2]In the original work $r = n$, but there are theoretical reasons for believing that $r \sim \sqrt{2n}$ is big enough [3], plus there is empirical evidence that much smaller values work in practice.

### References

[1] N. Alon and V.D. Milman. $\lambda_1$, isoperimetric inequalities for graphs, and superconcentrators. *Journal of Combinatorial Theory, Series B*, 38:73–88, 1985.

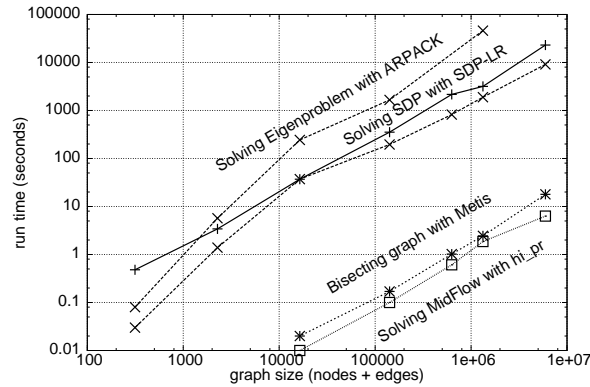

Figure 6: Run time scaling on subsets of the Yahoo IM graph. Finding Spectral and SDP embeddings with `ARPACK` and `SDP-LR` requires about the same amount of time, while MidFlow rounding with `hi_pr` is about 1000 times faster.

[2] Samuel Burer and Renato D.C. Monteiro. A nonlinear programming algorithm for solving semidefinite programs via low-rank factorization. *Mathematical Programming (series B)*, 95(2):329–357, 2003.

[3] Samuel Burer and Renato D.C. Monteiro. Local minima and convergence in low-rank semidefinite programming. Technical report, Department of Management Sciences, University of Iowa, September 2003.

[4] Boris V. Cherkassky and Andrew V. Goldberg. On implementing the push-relabel method for the maximum flow problem. *Algorithmica*, 19(4):390–410, 1997.

[5] F. Chung and L. Lu. Average distances in random graphs with given expected degree sequences. *Proceedings of National Academy of Science*, 99:15879–15882, 2002.

[6] W.E. Donath and A. J. Hoffman. Lower bounds for partitioning of graphs. *IBM J. Res. Develop.*, 17:420–425, 1973.

[7] Peter G. Doyle and J. Laurie Snell. Random walks and electric networks, 1984. Mathematical Association of America; now available under the GPL.

[8] C.M. Fiduccia and R.M. Mattheyses. A linear time heuristic for improving network partitions. In *Design Automation Conference*, pages 175–181, 1982.

[9] Michel X. Goemans and David P. Williamson. Improved Approximation Algorithms for Maximum Cut and Satisfiability Problems Using Semidefinite Programming. *J. Assoc. Comput. Mach.*, 42:1115–1145, 1995.

[10] Stephen Guattery and Gary L. Miller. On the quality of spectral separators. *SIAM Journal on Matrix Analysis and Applications*, 19(3):701–719, 1998.

[11] C. Helmberg. Numerical evaluation of sbmethod. *Math. Programming*, 95(2):381–406, 2003.

[12] Bruce Hendrickson and Robert W. Leland. A multi-level algorithm for partitioning graphs. In *Supercomputing*, 1995.

[13] Kevin Lang and Satish Rao. A flow-based method for improving the expansion or conductance of graph cuts. In *Integer Programming and Combinatorial Optimization*, pages 325–337, 2003.

[14] Umesh V. Vazirani Sanjeev Arora, Satish Rao. Expander flows, geometric embeddings and graph partitioning. In *STOC*, pages 222–231, 2004.

[15] Jianbo Shi and Jitendra Malik. Normalized cuts and image segmentation. *IEEE Transactions on Pattern Analysis and Machine Intelligence*, 22(8):888–905, 2000.

[16] Daniel A. Spielman and Shang-Hua Teng. Spectral partitioning works: Planar graphs and finite element meshes. In *FOCS*, pages 96–105, 1996.
